# Semi-Supervised Multitask Learning

**Qiuhua Liu, Xuejun Liao, and Lawrence Carin**
Department of Electrical and Computer Engineering
Duke University
Durham, NC 27708-0291, USA

## Abstract

A semi-supervised multitask learning (MTL) framework is presented, in which $M$ parameterized semi-supervised classifiers, each associated with one of $M$ partially labeled data manifolds, are learned jointly under the constraint of a soft-sharing prior imposed over the parameters of the classifiers. The unlabeled data are utilized by basing classifier learning on neighborhoods, induced by a Markov random walk over a graph representation of each manifold. Experimental results on real data sets demonstrate that semi-supervised MTL yields significant improvements in generalization performance over either semi-supervised single-task learning (STL) or supervised MTL.

## 1 Introduction

Supervised learning has proven an effective technique for learning a classifier when the quantity of labeled data is large enough to represent a sufficient sample from the true labeling function. Unfortunately, a generous provision of labeled data is often not available since acquiring the label of a datum is expensive in many applications. A classifier supervised by a limited amount of labeled data is known to generalize poorly even if it produces zero training errors. There has been much recent work on improving the generalization of classifiers based on using information sources beyond the labeled data. These studies fall into two major categories: (i) semi-supervised learning [9, 12, 15, 10] and (ii) multitask learning (MTL) [3, 1, 13]. The former employs the information from the data manifold, in which the manifold information provided by the usually abundant unlabeled data is exploited, while the latter leverages information from related tasks.

In this paper we attempt to integrate the benefits offered by semi-supervised learning and MTL, by proposing *semi-supervised multitask learning*. The semi-supervised MTL framework consists of $M$ semi-supervised classifiers coupled by a joint prior distribution over the parameters of all classifiers. Each classifier provides the solution for a partially labeled data classification task. The solutions for the $M$ tasks are obtained simultaneously under the unified framework.

Existing semi-supervised algorithms are often not directly amenable to MTL extensions. Transductive algorithms directly operate on labels. Since the label is a local property of the associated data point, information sharing must be performed at the level of data locations, instead of at the task level. The inductive algorithm in [10] employs a data-dependent prior to encode manifold information. Since the information transferred from related tasks is also often represented by a prior, the two priors will compete and need be balanced; moreover, this precludes a Dirichlet process [6] or its variants to represent the sharing prior across tasks, because the base distribution of a Dirichlet process cannot be dependent on any particular manifold.

We develop a new semi-supervised formulation, which enjoys several nice properties that make the formulation immediately amenable to an MTL extension. First, the formulation has a parametric classifier built for each task, thus multitask learning can be performed efficiently at the task level, using the parameters of the classifiers. Second, the formulation encodes the manifold information

of each task inside the associated likelihood function, sparing the prior for exclusive use by the information from related tasks. Third, the formulation lends itself to a Dirichlet process, allowing the tasks to share information in a complex manner.

The new semi-supervised formulation is used as a key component of our semi-supervised MTL framework. In the MTL setting, we have $M$ partially labeled data manifolds, each defining a classification task and involving design of a semi-supervised classifier. The $M$ classifiers are designed simultaneously within a unified sharing structure. The key component of the sharing structure is a soft variant of the Dirichlet process (DP), which implements a soft-sharing prior over the parameters of all classifiers. The soft-DP retains the clustering property of DP and yet does not require exact sharing of parameters, which increases flexibility and promotes robustness in information sharing.

## 2 Parameterized Neighborhood-Based Classification

The new semi-supervised formulation, termed *parameterized neighborhood-based classification (PNBC)*, represents the class probability of a data point by mixing over all data points in the neighborhood, which is formed via Markov random walk over a graph representation of the manifold.

### 2.1 Neighborhoods Induced by Markov Random Walk

Let $G = (\mathcal{X}, \mathbf{W})$ be a weighted graph such that $\mathcal{X} = \{\mathbf{x}_1, \mathbf{x}_2, \cdots, \mathbf{x}_n\}$ is a set of vertices that coincide with the data points in a finite data manifold, and $\mathbf{W} = [w_{ij}]_{n \times n}$ is the affinity matrix with the $(i, j)$-th element $w_{ij}$ indicating the immediate affinity between data points $\mathbf{x}_i$ and $\mathbf{x}_j$. We follow [12, 15] to define $w_{ij} = \exp(-0.5 \|\mathbf{x}_i - \mathbf{x}_j\|^2 / \sigma_i^2)$, where $\|\cdot\|$ is the Euclidean norm and $\sigma_{ij} > 0$.

A Markov random walk on graph $G = (\mathcal{X}, \mathbf{W})$ is characterized by a matrix of one-step transition probabilities $\mathbf{A} = [a_{ij}]_{n \times n}$, where $a_{ij}$ is the probability of transiting from $\mathbf{x}_i$ to $\mathbf{x}_j$ via a single step and is given by $a_{ij} = \frac{w_{ij}}{\sum_{k=1}^{n} w_{ik}}$ [4]. Let $\mathbf{B} = [b_{ij}]_{n \times n} = \mathbf{A}^t$. Then $(i, j)$-th element $b_{ij}$ represents the probability of transiting from $\mathbf{x}_i$ to $\mathbf{x}_j$ in $t$ steps.

Data point $\mathbf{x}_j$ is said to be a $t$-step neighbor of $\mathbf{x}_i$ if $b_{ij} > 0$. The $t$-step neighborhood of $\mathbf{x}_i$, denoted as $\mathcal{N}_t(\mathbf{x}_i)$, is defined by all $t$-step neighbors of $\mathbf{x}_i$ along with the associated $t$-step transition probabilities, i.e., $\mathcal{N}_t(\mathbf{x}_i) = \{(\mathbf{x}_j, b_{ij}) : b_{ij} > 0, \mathbf{x}_j \in \mathcal{X}\}$. The appropriateness of a $t$-step neighborhood depends on the right choice of $t$. A rule of choosing $t$ is given in [12], based on maximizing the margin of the associated classifier on both labeled and unlabeled data points.

The $\sigma_i$ in specifying $w_{ij}$ represents the step-size (distance traversed in a single step) for $\mathbf{x}_i$ to reach its immediate neighbor, and we have used a distinct $\sigma$ for each data point. Location-dependent step-sizes allow one to account for possible heterogeneities in the data manifold — at locations with dense data distributions a small step-size is suitable, while at locations with sparse data distributions a large step-size is appropriate. A simple choice of heterogeneous $\sigma$ is to let $\sigma_i$ be related to the distance between $\mathbf{x}_i$ and close-by data points, where closeness is measured by Euclidean distance. Such a choice ensures each data point is immediately connected to some neighbors.

### 2.2 Formulation of the PNBC Classifier

Let $p^*(y_i | \mathbf{x}_i, \boldsymbol{\theta})$ be a base classifier parameterized by $\boldsymbol{\theta}$, which gives the probability of class label $y_i$ of data point $\mathbf{x}_i$, given $\mathbf{x}_i$ alone (which is a zero-step neighborhood of $\mathbf{x}_i$). The base classifier can be implemented by any parameterized probabilistic classifier. For binary classification with $y \in \{-1, 1\}$, the base classifier can be chosen as logistic regression with parameters $\boldsymbol{\theta}$, which expresses the conditional class probability as

$$p^*(y_i | \mathbf{x}_i, \boldsymbol{\theta}) = [1 + \exp(-y_i \boldsymbol{\theta}^T \mathbf{x}_i)]^{-1} \tag{1}$$

where a constant element 1 is assumed to be prefixed to each $\mathbf{x}$ (the prefixed $\mathbf{x}$ is still denoted as $\mathbf{x}$ for notational simplicity), and thus the first element in $\boldsymbol{\theta}$ is a bias term.

Let $p(y_i | \mathcal{N}_t(\mathbf{x}_i), \boldsymbol{\theta})$ denote a neighborhood-based classifier parameterized by $\boldsymbol{\theta}$, representing the probability of class label $y_i$ for $\mathbf{x}_i$, given the neighborhood of $\mathbf{x}_i$. The PNBC classifier is defined as a mixture

$$p(y_i | \mathcal{N}_t(\mathbf{x}_i), \boldsymbol{\theta}) = \sum_{j=1}^{n} b_{ij} \, p^*(y_i | \mathbf{x}_j, \boldsymbol{\theta}) \tag{2}$$

where the $j$-th component is the base classifier applied to $(\mathbf{x}_j, y_i)$ and the associated mixing proportion is defined by the probability of transiting from $\mathbf{x}_i$ to $\mathbf{x}_j$ in $t$ steps. Since the magnitude of $b_{ij}$ automatically determines the contribution of $\mathbf{x}_j$ to the mixture, we let index $j$ run over the entire $\mathcal{X}$ for notational simplicity.

The utility of unlabeled data in (2) is conspicuous — in order for $\mathbf{x}_i$ to be labeled $y_i$, each neighbor $\mathbf{x}_j$ must be labeled consistently with $y_i$, with the strength of consistency proportional to $b_{ij}$; in such a manner, $y_i$ implicitly propagates over the neighborhood of $\mathbf{x}_i$. By taking neighborhoods into account, it is possible to obtain an accurate estimate of $\boldsymbol{\theta}$, based on a small amount of labeled data. The over-fitting problem associated with limited labeled data is ameliorated in the PNBC formulation, through enforcing consistent labeling over each neighborhood.

Let $\mathcal{L} \subseteq \{1, 2, \cdots, n\}$ denote the index set of labeled data in $\mathcal{X}$. Assuming the labels are conditionally independent, we write the neighborhood-conditioned likelihood function

$$p\big(\{y_i, i \in \mathcal{L}\}|\{\mathcal{N}_t(\mathbf{x}_i) : i \in \mathcal{L}\}, \boldsymbol{\theta}\big) = \prod_{i \in \mathcal{L}} p(y_i|\mathcal{N}_t(\mathbf{x}_i), \boldsymbol{\theta}) = \prod_{i \in \mathcal{L}} \sum_{j=1}^{n} b_{ij}\, p^*(y_i|\mathbf{x}_j, \boldsymbol{\theta}) \quad (3)$$

## 3 The Semi-Supervised MTL Framework

### 3.1 The sharing prior

Suppose we are given $M$ tasks, defined by $M$ partially labeled data sets

$$\mathcal{D}_m = \{\mathbf{x}_i^m : i = 1, 2, \cdots, n_m\} \cup \{y_i^m : i \in \mathcal{L}_m\}$$

for $m = 1, \cdots, M$, where $y_i^m$ is the class label of $\mathbf{x}_i^m$ and $\mathcal{L}_m \subset \{1, 2, \cdots, n_m\}$ is the index set of labeled data in task $m$. We consider $M$ PNBC classifiers, parameterized by $\boldsymbol{\theta}_m$, $m = 1, \cdots, M$, with $\boldsymbol{\theta}_m$ responsible for task $m$. The $M$ classifiers are not independent but coupled by a prior joint distribution over their parameters

$$p(\boldsymbol{\theta}_1, \cdots, \boldsymbol{\theta}_M) = \prod_{m=1}^{M} p(\boldsymbol{\theta}_m|\boldsymbol{\theta}_1, \cdots, \boldsymbol{\theta}_{m-1}) \quad (4)$$

with the conditional distributions in the product defined by

$$p(\boldsymbol{\theta}_m|\boldsymbol{\theta}_1, \cdots, \boldsymbol{\theta}_{m-1}) = \frac{1}{\alpha + m - 1}\big[\alpha p(\boldsymbol{\theta}_m|\Upsilon) + \sum_{l=1}^{m-1} N(\boldsymbol{\theta}_m; \boldsymbol{\theta}_l, \eta^2 \mathbf{I})\big] \quad (5)$$

where $\alpha > 0$, $p(\boldsymbol{\theta}_m|\Upsilon)$ is a base distribution parameterized by $\Upsilon$, $N(\,\cdot\,; \boldsymbol{\theta}_l, \eta^2 \mathbf{I})$ is a normal distribution with mean $\boldsymbol{\theta}_l$ and covariance matrix $\eta^2 \mathbf{I}$. As discussed below, the prior in (4) is linked to Dirichlet processes and thus is more general than a parametric prior, as used, for example, in [5].

Each normal distribution represents the prior transferred from a previous task; it is the meta-knowledge indicating how the present task should be learned, based on the experience with a previous task. It is through these normal distributions that information sharing between tasks is enforced. Taking into account the data likelihood, unrelated tasks cannot share since they have dissimilar solutions and forcing them to share the same solution will decrease their respective likelihood; whereas, related tasks have close solutions and sharing information helps them to find their solutions and improve their data likelihoods.

The base distribution represents the baseline prior, which is exclusively used when there are no previous tasks available, as is seen from (5) by setting $m = 1$. When there are $m - 1$ previous tasks, one uses the baseline prior with probability $\frac{\alpha}{\alpha+m-1}$, and uses the prior transferred from each of the $m - 1$ previous tasks with probability $\frac{1}{\alpha+m-1}$. The $\alpha$ balances the baseline prior and the priors imposed by previous tasks. The role of baseline prior decreases as $m$ increases, which is in agreement with our intuition, since the information from previous tasks increase with $m$.

The formulation in (5) is suggestive of the polya urn representation of a Dirichlet process (DP) [2]. The difference here is that we have used a normal distribution to replace Dirac delta in Dirichlet processes. Since $N(\boldsymbol{\theta}_m|\boldsymbol{\theta}_l, \eta^2 \mathbf{I})$ approaches Dirac delta $\delta(\boldsymbol{\theta}_m - \boldsymbol{\theta}_l)$ as $\eta^2 \to 0$, we recover the Dirichlet process in the limit case when limit case when $\eta^2 \to 0$.

The motivation behind the formulation in (5) is twofold. First, a normal distribution can be regarded as a soft version of the Dirac delta. While the Dirac delta requires two tasks to have exactly the same $\boldsymbol{\theta}$ when sharing occurs, the soft delta only requires sharing tasks to have similar $\boldsymbol{\theta}$'s. The

soft sharing may therefore be more consistent with situations in practical applications. Second, the normal distribution is analytically more appealing than the Dirac delta and allows simple *maximum a posteriori* (MAP) solutions. This is an attractive property considering that most classifiers do not have conjugate priors for their parameters and Bayesian learning cannot be performed exactly.

Under the sharing prior in (4), the current task is equally influenced by each previous task but is influenced unevenly by future tasks — a distant future task has less influence than a near future task. The ordering of the tasks imposed by (4) may in principle affect performance, although we have not found this to be an issue in the experimental results. Alternatively, one may obtain a sharing prior that does not depend on task ordering, by modifying (5) as

$$p(\boldsymbol{\theta}_m|\boldsymbol{\theta}_{-m}) = \tfrac{1}{\alpha+M-1}\big[\alpha p(\boldsymbol{\theta}_m|\Upsilon) + \textstyle\sum_{l\neq m} N(\boldsymbol{\theta}_m;\boldsymbol{\theta}_l,\eta^2\mathbf{I})\big] \tag{6}$$

where $\boldsymbol{\theta}_{-m} = \{\boldsymbol{\theta}_1,\cdots,\boldsymbol{\theta}_M\}\setminus\{\boldsymbol{\theta}_m\}$. The prior joint distribution of $\{\boldsymbol{\theta}_1,\cdots,\boldsymbol{\theta}_M\}$ associated with the full conditionals in (6) is not analytically available, nether is the corresponding posterior joint distribution, which causes technical difficulties in performing MAP estimation.

## 3.2 Maximum A Posteriori (MAP) Estimation

Assuming that, given $\{\boldsymbol{\theta}_1,\cdots,\boldsymbol{\theta}_M\}$, the class labels of different tasks are conditionally independent, the joint likelihood function over all tasks can be written as

$$p\big(\{y_i^m, i \in \mathcal{L}_m\}_{m=1}^M | \{\mathcal{N}_t(\mathbf{x}_i^m) : i \in \mathcal{L}_m\}_{m=1}^M, \{\boldsymbol{\theta}_m\}_{m=1}^M\big)$$
$$= \textstyle\prod_{m=1}^M \prod_{i\in\mathcal{L}_m} \sum_{j=1}^{n_m} b_{ij}^m\, p^*(y_i^m|\mathbf{x}_j^m,\boldsymbol{\theta}_m) \tag{7}$$

where the $m$-th term in the product is taken from (3), with the superscript $m$ indicating the task index. Note that the neighborhoods are built for each task independently of other tasks, thus a random walk is always restricted to the same task (the one where the starting data point belongs) and can never traverse multiple tasks. From (4), (5), and (7), one can write the logarithm of the joint *posterior* of $\{\boldsymbol{\theta}_1,\cdots,\boldsymbol{\theta}_M\}$, up to a constant translation that does not depend on $\{\boldsymbol{\theta}_1,\cdots,\boldsymbol{\theta}_M\}$,

$$\ell_{\mathrm{MAP}}(\boldsymbol{\theta}_1,\cdots,\boldsymbol{\theta}_M) = \ln p\big(\{\boldsymbol{\theta}_m\}_{m=1}^M | \{y_i^m, i \in \mathcal{L}_m\}_{m=1}^M, \{\mathcal{N}_t(\mathbf{x}_i^m) : i \in \mathcal{L}_m\}_{m=1}^M\big)$$
$$= \textstyle\sum_{m=1}^M \big\{ \ln\big[\alpha p(\boldsymbol{\theta}_m|\Upsilon) + \sum_{l=1}^{m-1} N(\boldsymbol{\theta}_m;\boldsymbol{\theta}_l,\eta^2\mathbf{I})\big] + \sum_{i\in\mathcal{L}_m} \ln\sum_{j=1}^{n_m} b_{ij}^m p^*(y_i^m|\mathbf{x}_j^m,\boldsymbol{\theta}_m)\big\} \tag{8}$$

We seek the parameters $\{\boldsymbol{\theta}_1,\cdots,\boldsymbol{\theta}_M\}$ that maximize the log-posterior, which is equivalent to simultaneously maximizing the prior in (4) and the likelihood function in (7). As seen from (5), the prior tends to have similar $\boldsymbol{\theta}$'s across tasks (similar $\boldsymbol{\theta}$'s increase the prior); however sharing between unrelated tasks is discouraged, since each task requires a distinct $\boldsymbol{\theta}$ to make its likelihood large. As a result, to make the prior and the likelihood large at the same time, one must let related tasks have similar $\boldsymbol{\theta}$'s. Although any optimization techniques can be applied to maximize the objective function (8), expectation maximization (EM) is particularly suitable, since the objective function involves summations under the logarithmic operation. To conserve space the algorithmic details are omitted here.

Utilization of the manifold information and the information from related tasks has greatly reduced the hypothesis space. Therefore, point MAP estimation in semi-supervised MTL will not suffer as much from overfitting as in supervised STL. This argument will be supported by the experimental results in Section 4.2, where semi-supervised MTL outperforms both supervised MTL and supervised STL, although the former is based on MAP and the latter two are based on Bayesian learning.

With MAP estimation, one obtains the parameters of the base classifier in (1) for each task, which can be employed to predict the class label of any data point in the associated task, regardless of whether the data point has been seen during training. In the special case when predictions are desired only for the unlabeled data points seen during training (transductive learning), one can alternatively employ the PNBC classifier in (2) to perform the predictions.

## 4  Experimental Results

First we consider semi-supervised learning on a single task and establish the competitive performance of the PNBC in comparison with existing semi-supervised algorithms. Then we demonstrate the performance improvements achieved by semi-supervised MTL, relative to semi-supervised STL and supervised MTL. Throughout this section, the base classifier in (1) is logistic regression.

## 4.1 Performance of the PNBC on a Single Task

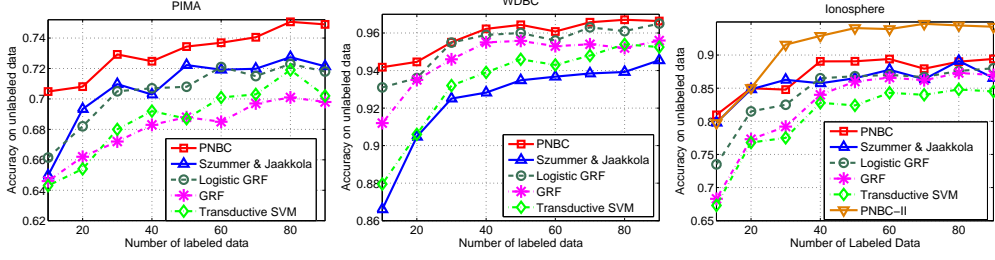

Figure 1: Transductive results of the PNBC. The horizontal axis is the size of $\mathcal{X}_L$.

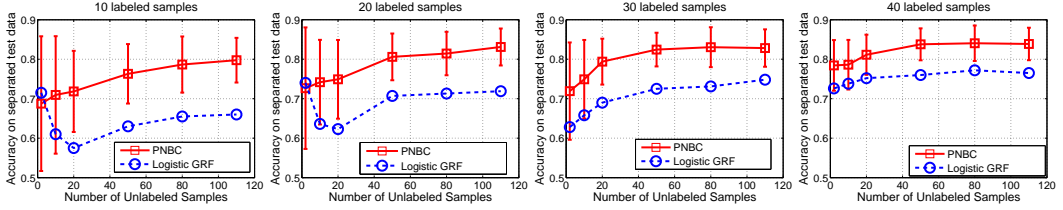

Figure 2: Inductive results of the PNBC on Ionosphere. The horizontal axis is the size of $\mathcal{X}_U$.

The PNBC is evaluated on three benchmark data sets – Pima Indians Diabetes Database (PIMA), Wisconsin Diagnostic Breast Cancer (WDBC) data, and Johns Hopkins University Ionosphere database (Ionosphere), which are taken from the UCI machine learning repository [11]. The evaluation is performed in comparison to four existing semi-supervised learning algorithms, namely, the transductive SVM [9], the algorithm of Szummer & Jaakkola [12], GRF [15], and Logistic GRF [10]. The performance is evaluated in terms of classification accuracy, defined as the ratio of the number of correctly classified data over the total number of data being tested.

We consider two testing modes: transductive and inductive. In the transductive mode, the test data are the unlabeled data that are used in training the semi-supervised algorithms; in the inductive mode, the test data are a set of holdout data unseen during training. We follow the same procedures as used in [10] to perform the experiments. Denote by $\mathcal{X}$ any of the three benchmark data sets and $\mathcal{Y}$ the associated set of class labels. In the transductive mode, we randomly sample $\mathcal{X}_L \subset \mathcal{X}$ and assume the associated class labels $\mathcal{Y}_L$ are available; the semi-supervised algorithms are trained by $\mathcal{X} \cup \mathcal{Y}_L$ and tested on $\mathcal{X} \setminus \mathcal{X}_L$. In the inductive mode, we randomly sample two disjoint data subsets $\mathcal{X}_L \subset \mathcal{X}$ and $\mathcal{X}_U \subset \mathcal{X}$, and assume the class labels $\mathcal{Y}_L$ associated with $\mathcal{X}_L$ are available; the semi-supervised algorithms are trained by $\mathcal{X}_L \cup \mathcal{Y}_L \cup \mathcal{X}_U$ and tested on 200 data randomly sampled from $\mathcal{X} \setminus (\mathcal{X}_L \cup \mathcal{X}_U)$.

The comparison results are summarized in Figures 1 and 2, where the results of the PNBC and the algorithm of Szummer & Jaakkola are calculated by us, and the results of remaining algorithms are cited from [10]. The algorithm of Szummer & Jaakkola [12] and the PNBC use $\sigma_i = \min_j \|\mathbf{x}_i - \mathbf{x}_j\|/3$ and $t = 100$; learning of the PNBC is based on MAP estimation. Each curve in the figures is a result averaged from $T$ independent trials, with $T = 20$ for the transductive results and $T = 50$ for the inductive results. In the inductive case, the comparison is between the proposed algorithm and the Logistic GRF, as the others are transductive algorithms.

For the PNBC, we can either use the base classifier in (1) or the PNBC classifier in (2) to predict the labels of unlabeled data seen in training (the transductive mode). In the inductive mode, however, the $\{b_{ij}\}$ are not available for the test data (unseen in training) since they are not in the graph representation, therefore we can only employ the base classifier. In the legends of Figures 1 and 2, a suffix "II" to PNBC indicates that the PNBC classifier in (2) is employed in testing; when no suffix is attached, the base classifier is employed in testing.

Figures 1 and 2 show that the PNBC outperforms all the competing algorithms in general, regardless of the number of labeled data points. The improvements are particularly significant on PIMA and

Ionosphere. As indicated in Figure 1(c), employing manifold information in testing by using (2) can improve classification accuracy in the transductive learning case. The margin of improvements achieved by the PNBC in the inductive learning case is striking and encouraging — as indicated by the error bars in Figure 2, the PNBC significantly outperforms Logistic GRF in almost all individual trials. Figure 2 also shows that the advantage of the PNBC becomes more conspicuous with decreasing amount of labeled data considered during training.

## 4.2 Performance of the Semi-Supervised MTL Algorithm

We compare the proposed semi-supervised MTL against: (a) semi-supervised single-task learning (STL), (b) supervised MTL, (c) supervised STL, (d) supervised pooling; STL refers to designing $M$ classifiers independently, each for the corresponding task, and pooling refers to designing a single classifier based on the data of all tasks. Since we have evaluated the PNBC in Section 4.1 and established its effectiveness, we will not repeat the evaluation here and employ PNBC as a representative semi-supervised learning algorithm in semi-supervised STL. To replicate the experiments in [13], we employ AUC as the performance measure, where AUC stands for area under the receiver operation characteristic (ROC) curve [7].

The basic setup of the semi-supervised MTL algorithm is as follows. The tasks are ordered as they are when the data are provided to the experimenter (we have randomly permuted the tasks and found the performance does not change much). A separate $t$-neighborhood is employed to represent the manifold information (consisting of labeled and unlabeled data points) for each task, where the step-size at each data point is one third of the shortest distance to the remaining points and $t$ is set to half the number of data points. The base prior $p(\boldsymbol{\theta}_m|\boldsymbol{\Upsilon}) = N(\boldsymbol{\theta}_m; \mathbf{0}, \upsilon^2 \mathbf{I})$ and the soft delta is $N(\boldsymbol{\theta}_m; \boldsymbol{\theta}_l, \eta^2 \mathbf{I})$, where $\upsilon = \eta = 1$. The $\alpha$ balancing the base prior and the soft delta's is 0.3. These settings represent the basic intuition of the experimenter; they have not been tuned in any way and therefore do not necessarily represent the best settings for the semi-supervised MTL algorithm.

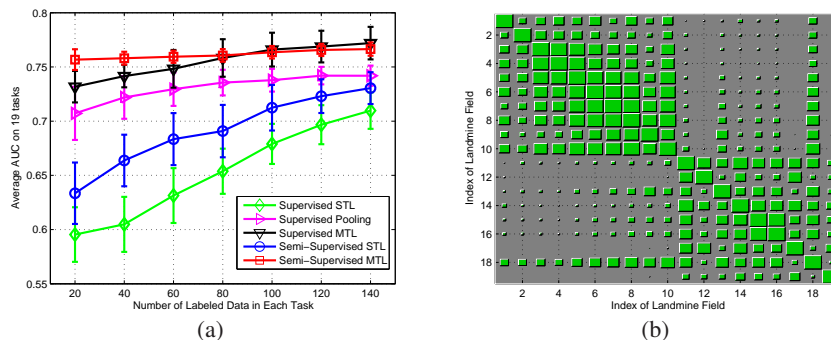

Figure 3: (a) Performance of the semi-supervised MTL algorithm on landmine detection, in comparison to the remaining five algorithms. (b) The Hinton diagram of between-task similarity when there are 140 labeled data in each task.

**Landmine Detection**    First we consider the remote sensing problem considered in [13], based on data collected from real landmines. In this problem, there are a total of 29 sets of data, collected from various landmine fields. Each data point is represented by a 9-dimensional feature vector extracted from radar images. The class label is binary (mine or false mine). The data are available at http://www.ee.duke.edu/~lcarin/LandmineData.zip.

Each of the 29 data sets defines a task, in which we aim to find landmines with a minimum number of false alarms. To make the results comparable to those in [13], we follow the authors there and take data sets 1-10 and 16-24 to form 19 tasks. Of the 19 selected data sets, 1-10 are collected at foliated regions and 11-19 are collected at regions that are bare earth or desert. Therefore we expect two dominant clusters of tasks, corresponding to the two different types of ground surface conditions.

To replicate the experiments in [13], we perform 100 independent trials, in each of which we randomly select a subset of data for which labels are assumed available, train the semi-supervised MTL

and semi-supervised STL classifiers, and test the classifiers on the remaining data. The AUC averaged over the 19 tasks is presented in Figure 3(a), as a function of the number of labeled data, where each curve represents the mean calculated from the 100 independent trials and the error bars represent the corresponding standard deviations. The results of supervised STL, supervised MTL, and supervised pooling are cited from [13].

Semi-supervised MTL clearly yields the best results up to 80 labeled data points; after that supervised MTL catches up but semi-supervised MTL still outperforms the remaining three algorithms by significant margins. In this example semi-supervised MTL seems relatively insensitive to the amount of labeled data; this may be attributed to the doubly enhanced information provided by the data manifold plus the related tasks, which significantly augment the information available in the limited labeled data. The superiority of supervised pooling over supervised STL on this dataset suggests that there are significant benefits offered by sharing across the tasks, which partially explains why supervised MTL eventually catches up with semi-supervised MTL.

We plot in Figure 3(b) the Hinton diagram [8] of the between-task sharing matrix (an average over the 100 trials) found by the semi-supervised MTL when there are 140 labeled data in each task. The $(m, l)$-th element of similarity matrix is equal to $\exp(-\frac{\|\boldsymbol{\theta}_m - \boldsymbol{\theta}_l\|^2}{2})$ (normalized such that the maximum element is one), which is represented by a square in the Hinton diagram, with a larger square indicating a larger value of the corresponding element. As seen from Figure 3(b), there is a dominant sharing among tasks 1-10 and another dominant sharing among tasks 11-19. Recall from the beginning of the section that data sets 1-10 are from foliated regions and data sets 11-19 are from regions that are bare earth or desert. Therefore, the sharing is in agreement with the similarity between tasks.

**Art Images Retrieval**   We now consider the problem of art image retrieval [14, 13], in which we have a library of 642 art images and want to retrieve the images based on a user's preference. The preference of each user is available on a subset of images, therefore the objective is to learn the preference of each user based on a subset of training examples. Each image is represented by a vector of features and a user's rating is represented by a binary label (like or dislike). The users' preferences are collected in a web-based survey, which can be found at http://honolulu.dbs.informatik.uni-muenchen.de:8080/paintings/index.jsp.

We consider the same 69 users as considered in [13], who each rated more than 100 images. The preference prediction for each user is treated as a task, with the associated set of ground truth data defined by the images rated by the user. These 69 tasks are used in our experiment to evaluate the performance of semi-supervised MTL. Since two users may give different ratings to exactly the same image, pooling the tasks together can lead to multiple labels for the same data point. For this reason, we exclude supervised pooling and semi-supervised pooling in the performance comparison.

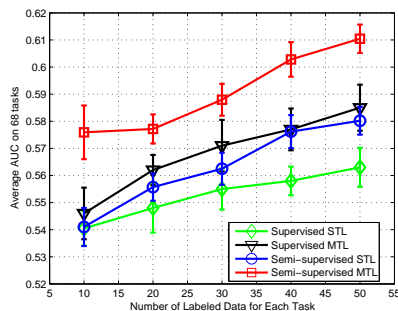

Figure 4: Performance of the semi-supervised MTL algorithm on art image retrieval, in comparison to the remaining three algorithms.

Following [13], we perform 50 independent trials, in each of which we randomly select a subset of images rated by each user, train the semi-supervised MTL and semi-supervised STL classifiers, and test the classifiers on the remaining images. The AUC averaged over the 69 tasks is presented in Figure 4, as a function of the number of labeled data (rated images), where each curve represents

the mean calculated from the 50 independent trials and the error bars represent the corresponding standard deviations. The results of supervised STL and supervised MTL are cited from [13].

Semi-supervised MTL performs very well, improving upon results of the three other algorithms by significant margins in almost all individual trials (as seen from the error bars). It is noteworthy that the performance improvement achieved by semi-supervised MTL over semi-supervised STL is larger than corresponding improvement achieved by supervised MTL over supervised STL. The greater improvement demonstrates that unlabeled data can be more valuable when used along with multitask learning. The additional utility of unlabeled data can be attributed to its role in helping to find the appropriate sharing between tasks.

## 5 Conclusions

A framework has been proposed for performing semi-supervised multitask learning (MTL). Recognizing that existing semi-supervised algorithms are not conveniently extended to an MTL setting, we have introduced a new semi-supervised formulation to allow a direct MTL extension. We have proposed a soft sharing prior, which allows each task to robustly borrow information from related tasks and is amenable to simple point estimation based on *maximum a posteriori*. Experimental results have demonstrated the superiority of the new semi-supervised formulation as well as the additional performance improvement offered by semi-supervised MTL. The superior performance of semi-supervised MTL on art image retrieval and landmine detection show that manifold information and the information from related tasks could play positive and complementary roles in real applications, suggesting that significant benefits can be offered in practice by semi-supervised MTL.

## References

[1] B. Bakker and T. Heskes. Task clustering and gating for Bayesian multitask learning. *Journal of Machine Learning Research*, pages 83–99, 2003.

[2] D. Blackwell and J. MacQueen. Ferguson distributions via polya urn schemes. *Annals of Statistics*, 1: 353–355, 1973.

[3] R. Caruana. Multitask learning. *Machine Learning*, 28:41–75, 1997.

[4] F. R. K. Chung. *Spectral Graph Theory*. American Mathematical Society, 1997.

[5] T. Evgeniou and M. Pontil. Regularized multi-task learning. In *Proc. 17th SIGKDD Conf. on Knowledge Discovery and Data Mining*, 2004.

[6] T. Ferguson. A Bayesian analysis of some nonparametric problems. *Annals of Statistics*, 1:209–230, 1973.

[7] J. Hanley and B. McNeil. The meaning and use of the area under a receiver operating characteristic (ROC) curve. *Radiology*, 143:29–36, 1982.

[8] G. E. Hinton and T. J. Sejnowski. Learning and relearning in boltzmann machines. In J. L. McClelland, D. E. Rumelhart, and the PDP Research Group, editors, *Parallel Distributed Processing: Explorations in the Microstructure of Cognition*, volume 1, pages 282–317. MIT Press, Cambridge, MA, 1986.

[9] T. Joachims. Transductive inference for text classification using support vector machines. In *Proc. 16th International Conf. on Machine Learning (ICML)*, pages 200–209. Morgan Kaufmann, San Francisco, CA, 1999.

[10] B. Krishnapuram, D. Williams, Y. Xue, A. Hartemink, L. Carin, and M. Figueiredo. On semi-supervised classification. In *Advances in Neural Information Processing Systems (NIPS)*, 2005.

[11] D.J. Newman, S. Hettich, C.L. Blake, and C.J. Merz. UCI repository of machine learning databases. *http://www.ics.uci.edu/~mlearn/MLRepository.html*, 1998.

[12] M. Szummer and T. Jaakkola. Partially labeled classification with markov random walks. In *Advances in Neural Information Processing Systems (NIPS)*, 2002.

[13] Y. Xue, X. Liao, L. Carin, and B. Krishnapuram. Multi-task learning for classification with dirichlet process priors. *Journal of Machine Learning Research (JMLR)*, 8:35–63, 2007.

[14] K. Yu, A. Schwaighofer, V. Tresp, W.-Y. Ma, and H.J. Zhang. Collaborative ensemble learning: Combining collaborative and content-based information filtering via hierarchical bayes. In *Proceedings of the 19th International Conference on Uncertainty in Artificial Intelligence (UAI 2003)*, 2003.

[15] X. Zhu, Z. Ghahramani, and J. Lafferty. Semi-supervised learning using gaussian fields and harmonic functions. In *The Twentieth International Conference on Machine Learning (ICML)*, 2003.
